# Synergies in learning words and their referents

**Mark Johnson**
Department of Computing
Macquarie University
Sydney, NSW 2109
Mark.Johnson@mq.edu.au

**Katherine Demuth**
Department of Linguistics
Macquarie University
Sydney, NSW 2109
Katherine.Demuth@mq.edu.au

**Michael Frank**
Department of Psychology
Stanford University
Palo Alto, CA 94305
mcfrank@mit.edu

**Bevan K. Jones**
School of Informatics
University of Edinburgh
10 Crichton Street, Edinburgh EH8 9AB, UK
B.K.Jones@sms.ed.ac.uk

## Abstract

This paper presents Bayesian non-parametric models that simultaneously learn to segment words from phoneme strings and learn the referents of some of those words, and shows that there is a synergistic interaction in the acquisition of these two kinds of linguistic information. The models themselves are novel kinds of Adaptor Grammars that are an extension of an embedding of topic models into PCFGs. These models simultaneously segment phoneme sequences into words and learn the relationship between non-linguistic objects to the words that refer to them. We show (i) that modelling inter-word dependencies not only improves the accuracy of the word segmentation but also of word-object relationships, and (ii) that a model that simultaneously learns word-object relationships and word segmentation segments more accurately than one that just learns word segmentation on its own. We argue that these results support an interactive view of language acquisition that can take advantage of synergies such as these.

## 1 Introduction

Conventional views of language acquisition often assume that human language learners initially use a single source of information to acquire one component of language, which they then use to leverage the acquisition of other linguistic components. For example, Kuhl [1] presents a standard "bootstrapping" view of early language acquisition in which successively more difficult tasks are addressed by learners, beginning with phoneme inventory and progressing to word segmentation and word learning. This view is also taken implicitly by, e.g., Graf Estes et al [2], who showed that infants were more successful in mapping novel objects to novel words after those words had been successfully segmented from the speech stream. We contrast this view with an "interactive" view of language acquisiion in which learners do not move from problem to problem, but instead attempt to learn all of the components of language at once. Computationally speaking, an interactive account views language acquisition as a joint inference problem for all components of language simultaneously, rather than a discrete sequence of inference problems for individual language components. (We are thus using "interactive" to refer to the way that language acquisition is formulated as an inference problem, rather than a specific mechanism or architecture as in [3]).

One advantage of an interactive approach is that it can take advantage of *synergies in acquisition*, i.e., situations where partial knowledge of several different aspects of language mutually aid their acquisition, i.e., where improvements in the acquisition of component $A$ also improves the acqui-

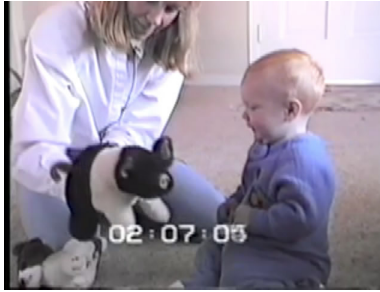

PIG|DOG   i △ z ▲ D △ & △ t ▲ D △ e ▲ p △ I △ g
$\underbrace{\phantom{p \, I \, g}}_{\text{PIG}}$

Figure 1: The photograph indicates non-linguistic context containing the (toy) pig and dog for the utterance *Is that the pig?*. Below that, we show the input provided to our models representing this utterance [8]. The objects in the non-linguistic context are indicated by the prefix "PIG|DOG", which is followed by the unsegmented phonemicised input. The possible word segmentation points are indicated by separators between the phonemes. The correct analysis of this input (which is not provided to the model) is depicted by blue annotations to this input. The correct word segmentation is indicated by the filled blue word separators, and the mapping between words and non-linguistic objects is indicated by the underbrace subscript.

sition of component $B$, and improvements in the acquisition of component $B$ also improves the acquisition of component $A$. An interactive approach can take advantage of both of these, while staged approach to activation where $A$ is learned before $B$ forgoes the ability to use knowledge of $B$ to help learn $A$.

In this paper we focus on the acquisition of two of the simpler aspects of language: (i) segmenting sentences into words (thereby identifying their pronunciations), and (ii) the relationship between words and the objects they refer to. We present a sequence of models for inferring (i) and (ii), and demonstrate synergistic interactions in learning. Specifically, we show that (i) modifying the model in a way that improves its word segmentation ability also improves its ability to identify the intended referents of utterances, and that (ii) incorporating a more sophisticated model of the relationship between words and the objects they refer to also improves the model's ability to segment words.

The acquisition of word pronunciations is viewed as a segmentation problem as follows. Following Elman [4] and Brent [5, 6], a corpus of child-directed speech is "phonemicised" by looking each word up in a pronouncing dictionary and concatenating those pronunciations. For example, the mother's utterance *Is that the pig* is mapped to the broad phonemic representation *Iz D&t D6 pIg* (in an ASCII-based broad phonemic encoding), which are then concatenated to form *IzD&tD6pIg*. The word segmentation task is to segment a corpus of such unsegmented utterance representations into words, thus identifying the pronunciations of the words in the corpus.

We study the acquisition of the relationship between words and the objects they refer to using the framework proposed by Frank et al [7]. Here each utterance in the corpus is labelled with the contextually-relevant objects that the speaker might be referring to. These are determined by inspecting videos of the utterance context. For example, in the context of Figure 1, the utterance would be labelled with the two contextually-relevant objects *PIG* and *DOG*. The learner's task is to identify which words, if any, in the utterance refer to each of these objects.

Jones et al [8] combined the word segmentation and word reference tasks into a single inference task, where the goal is to simultaneously segment the utterance into words, and to map a subset of the words of each utterance to the utterance's contextually-relevant objects. This is the task that we investigate in this paper.

The rest of this paper is structured as follows. The next section summarises previous work on word segmentation and learning the relationship between words and their referents. Section 3 introduces Adaptor Grammars, explains how they can be used for word segmentation and topic modelling, and presents the Adaptor Grammars that will be used in this paper. Section 4 presents experimental

results showing synergistic interactions between word segmentation and learning the relationship between words and the objects they refer to, while section 5 summarises and concludes the paper.

## 2 Previous work

Word segmentation has been studied using a wide variety of computational perspectives. Elman [4] and Brent [5, 6] introduced the basic word segmentation paradigm investigated here. Goldwater et al [9] introduced a non-parametric model of word segmentation based on Hierarchical Dirichlet Processes (HDPs) [10], and demonstrated that a bigram model, which captures dependencies between adjacent words, produces significantly more accurate segmentations than a unigram model, which assumes each word in a sentence is generated independently. Because the unigram model makes the "bag of words" assumption it has no way to capture inter-word dependencies. Because there are strong inter-word dependencies in real language, e.g., a noun like *ball* is very likely to be preceeded by determiners *the* or *a*, a unigram model tends to undersegment, e.g., misanalyse *the ball* as a single word. The bigram model, because it explicitly models and hence can "explain away" the dependency between *the* and *ball*, is more likely to correctly segment this example.

Johnson et al [11] introduced a generalisation of Probabilistic Context-Free Grammars (PCFGs) called Adaptor Grammars (AGs) as a framework for specifying HDPs for linguistic applications (because this paper relies heavily on AGs we describe them in more detail in section 3 below). Johnson [12] investigated AGs for word segmentation that capture a range of different kinds of generalisations. The unigram AG replicates the unigram segmentation model of Goldwater et al, and suffers from the same undersegmentation problems. It turns out that it is not possible to express Goldwater et al's bigram model as an AG, but a collocation AG, which is a HDP that generates a sentence as a sequence of collocations where each collocation is a sequence of words, captures similar inter-word dependencies and produces very similiar word segmentation results.

The acquisition of the mapping between words and the objects they refer to was studied by Frank et al [7]. They used a modified version of the LDA topic model [13] where the "topics" are contextually-relevant objects that words in the utterance can refer to, so the mapping from "topics" to words effectively specifies which words refer to these contextually-salient objects. Jones et al [8] integrated the Frank et al "topic" model of the word-object relationship with the unigram model of Goldwater et al to obtain a joint model that both performs word segmentation and also learns which words refer to which contextually-salient objects.

Johnson [14] explains how LDA topic models can be expressed as PCFGs. We use this reduction to express Frank et al models [7] of the word to object relationship as AGs which also incorporate Johnson's [12] models of word segmentation. The resulting AGs can express a wide range of joint HDP models of word segmentation and the word-object relationship, including the model proposed by Jones et al [8], as well as several generalisations.

## 3 Adaptor grammars for segmentation and word-object acquisition

This section provides an informal introduction to Adaptor Grammars (AGs) and how they can be used to express word segmentation and topic models, and presents the AGs for joint segmentation and acquisition of the word-object relationship. For more detail on the formal properties of AGs see [11], and for information on AG inference procedures see [15, 16].

### 3.1 Probabilistic Context-Free Grammars

Adaptor Grammars (AGs) are an extension of Probabilistic Context-Free Grammars (PCFGs), which we describe first. A *Context-Free Grammar* (CFG) $G = (N, W, R, S)$ consists of disjoint finite sets of *nonterminal symbols* $N$ and *terminal symbols* $W$, a finite set of *rules* $R$ of the form $A \rightarrow \alpha$ where $A \in N$ and $\alpha \in (N \cup W)^\star$, and a *start symbol* $S \in N$. (We assume there are no "$\epsilon$-rules" in $R$, i.e., we require that $|\alpha| \geq 1$ for each $A \rightarrow \alpha \in R$).

A CFG $G$ generates a set of finite, labelled, ordered trees $\mathcal{T}_X$ for each $X \in N \cup W$. If $X \in W$ (i.e., $X$ is a terminal) then $\mathcal{T}_X = \{X\}$, i.e., the singleton set consisting of a one-node tree labelled $X$. If $X \in N$ then $\mathcal{T}_X$ consists of all trees $t$ whose root node is labelled $X$, each leaf node's label is in

$W$, each non-leaf node's label is in $N$, and for each non-leaf node $x$ in $t$ with label $A \in N$ there is a rule $A \rightarrow \alpha \in R$ such that the sequence of labels of $x$'s children is $\alpha$. The set of strings generated by $G$ is the set of yields of $\mathcal{T}_S$, where the *yield* of a tree is sequence of its leaf nodes' labels.

A *Probabilistic Context-Free Grammar* PCFG is a quintuple $(N, W, R, S, \boldsymbol{\theta})$ where $(N, W, R, S)$ is a CFG and $\boldsymbol{\theta}$ is a vector of non-negative reals indexed by $R$ that satisfy $\sum_{\alpha \in R_A} \theta_{A \rightarrow \alpha} = 1$ for each $A \in N$, where $R_A = \{A \rightarrow \alpha : A \rightarrow \alpha \in R\}$ is the set of rules expanding $A$.

Informally, $\theta_{A \rightarrow \alpha}$ is the probability of a node labelled $A$ expanding to a sequence of nodes labelled $\alpha$, and the probability of a tree is the product of the probabilities of the rules used to construct each non-leaf node in it. More precisely, for each $X \in N \cup W$ a PCFG associates distributions $G_X$ over the trees $\mathcal{T}_X$ as follows:

If $X \in W$ (i.e., if $X$ is a terminal) then $G_X$ is the distribution that puts probability 1 on the single-node tree labelled $X$. If $X \in N$ (i.e., if $X$ is a nonterminal) then:

$$G_X \quad = \quad \sum_{X \rightarrow B_1 \ldots B_n \in R_X} \theta_{X \rightarrow B_1 \ldots B_n} \mathrm{TD}_X(G_{B_1}, \ldots, G_{B_n}) \tag{1}$$

where:

$$\mathrm{TD}_A(G_1, \ldots, G_n) \left( \overbrace{\underset{t_1 \ \ldots \ t_n}{X}} \right) \quad = \quad \prod_{i=1}^{n} G_i(t_i).$$

That is, $\mathrm{TD}_A(G_1, \ldots, G_n)$ is a distribution over $\mathcal{T}_A$ where each subtree $t_i$ is generated independently from $G_i$. The PCFG generates the distribution $G_S$ over the trees $\mathcal{T}_S$, where $S$ is the start symbol; the distribution over the strings it generates is obtained by marginalising over the trees.

In a Bayesian PCFG one puts Dirichlet priors $\mathrm{Dir}(\boldsymbol{\alpha})$ on the rule probability vector $\boldsymbol{\theta}$, such that there is one Dirichlet parameter $\alpha_{A \rightarrow \alpha}$ for each rule $A \rightarrow \alpha \in R$. In the "unsupervised" inference problem for a PCFG one is given a CFG, parameters $\boldsymbol{\alpha}$ for Dirichlet priors over the rule probabilities, and a corpus of strings. The task is to infer the corresponding posterior distribution over rule probabilities $\boldsymbol{\theta}$. Recently Bayesian inference algorithms for PCFGs have been described. Kurihara et al [17] describe a Variational Bayes algorithm for inferring PCFGs using a mean-field approximation, while Johnson et al [18] describe a Markov Chain Monte Carlo algorithm based on Gibbs sampling.

## 3.2 Modelling word-object reference using PCFGs

This section presents a novel encoding of a Frank et al [7] model for identifying word-object relationships as a PCFG. It is an adaptation of the reduction of LDA topic models to PCFGs given by Johnson [14]. That paper showed how to construct a PCFG that generates the same distribution over a collection of documents as an LDA model, and where Bayesian inference for the PCFG's rule probabilities yields the corresponding distributions as Bayesian inference of the corresponding LDA models. Because the Frank et al [7] model of the word-object relationship is very similiar to an LDA topic model, we can use the same techniques to design Bayesian PCFGs that infer word-object relationships.

The models we investigate in this paper assume that the words in a single sentence refer to at most one non-linguistic object (although it would be easy to relax this restriction). In this subsection we assume that the vocabulary $V$ (i.e., a set of words) is given, as is the set $O$ of objects that they can refer to. Let $O' = O \cup \{\emptyset\}$, where $\emptyset$ is a distinguished "null object" not in $O$, and let the nonterminals $N = \{S\} \cup \{A_o, B_o : o \in O'\}$, where $A_o$ and $B_o$ are nonterminals indexed by the $o \in O'$. Informally, a nonterminal $B_o$ expanding to word $w \in V$ indicates that $w$ refers to object $o$, while a $B_\emptyset$ expanding to $w$ indicates that $w$ is non-referential.

The set of objects in the non-linguistic context of an utterance is indicated by prefixing the utterance with a *context identifier* associated with those objects, such as "PIG|DOG" in Figure 1. A context identifier $c$ is a subset of $O'$ that contains $\emptyset$ (i.e., the null object is always in context). We assume we are given a (non-empty) set $C$ of context identifiers disjoint from $V$. Then the terminals of the

```
                          S
                          |
                        A_pig
                     ___/    \___
                  A_pig        B_pig
               ___/   \___       |
            A_pig      B_∅       pig
          __/   \__     |
       A_pig    B_∅    the
      _/   \_    |
  A_pig   B_∅   that
    |      |
PIG|DOG    is
```

Figure 2: A tree generated by the reference PCFG encoding a Frank et al [7] model of the word-object relationship. The yield of this tree corresponds to the sentence *Is that the pig*, and the context identifier is "PIG|DOG".

PCFG are $W = V \cup C$, and the rules $R$ of the PCFG are all instances of the following schemata:

$$
\begin{aligned}
&S \rightarrow A_o & &o \in O' \\
&A_o \rightarrow c & &c \in C, o \in c \\
&A_o \rightarrow A_o\, B_o & &o \in O' \\
&A_o \rightarrow A_o\, B_\emptyset & &o \in O' \\
&B_o \rightarrow w & &o \in O', w \in V
\end{aligned}
\tag{2}
$$

We call this the *reference* PCFG because it generates word-object reference pairs. An example of a tree generated by this grammar is shown in Figure 2. This grammar generates sentences consisting of a context identifier followed by a sequence of words; e.g. *PIG|DOG is that the pig*. Informally, the rule expanding S picks an object $o$ that the words in the object can refer to (if $o = \emptyset$ then all words in the sentence are non-referential). The first rule expanding $A_o$ ensures that $o$ is a member of that sentence's non-linguistic context, the second rule generates a $B_o$ that will ultimately generate a word $w$ (which we take to indicate that $w$ refers to $o$), while the third rule generates a word associated with the null object $\emptyset$.

A slightly more complicated PCFG, which we call the *reference1* grammar, can enforce the requirement that there is at most one referential word in each sentence. This constraint often holds in the simple sentences that appear in infant-directed speech (e.g., in *Is that the pig?*, the pig is only mentioned once).

$$
\begin{aligned}
&S \rightarrow S\, B_\emptyset & & \\
&S \rightarrow c & &c \in C \\
&S \rightarrow A_o\, B_o & &o \in O \\
&A_o \rightarrow c & &c \in C, o \in c \\
&A_o \rightarrow A_o\, B_\emptyset & &o \in O \\
&B_o \rightarrow w & &o \in O', w \in V
\end{aligned}
\tag{3}
$$

In this grammar the nonterminal labels function as states that record not just which object a referential word refers to, but also whether that referential word has been generated or not. Viewed top-down, the switch from S to $A_o$ indicates that a word from $B_o$ has just been generated (i.e., which we interpret as referring to object $o$). This object $o$ is passed down the $A_o$ chain generating words from $B_\emptyset$; the final expansion of $A_o \rightarrow c$ checks that $o$ is compatible with the context indicator $c$.

### 3.3 Adaptor grammars

This subsection briefly reviews adaptor grammars; for more detail see [11]. An *Adaptor Grammar* (AG) is a septuple $(N, W, R, S, \theta, A, C)$ consisting of a PCFG $(N, W, R, S, \theta)$ in which a subset $A \subseteq N$ of the nonterminals are identified as *adapted*, and where each adapted nonterminal $X \in A$ has an associated adaptor $C_X$. An *adaptor* $C_X$ for $X$ is a function that maps a distribution over trees $\mathcal{T}_X$ to a distribution over distributions over $\mathcal{T}_X$. In this paper we use two-parameter Poisson-Dirirchlet distributions as adaptors, so the corresponding predictive distributions are *Pitman-Yor Processes* (PYPs).

Just as for a PCFG, an AG defines distributions $G_X$ over trees $\mathcal{T}_X$ for each $X \in N \cup W$. If $X \in W$ or $X \notin A$ then $G_X$ is defined just as for a PCFG above, i.e., using (1). However, if $X \in A$ then $G_X$ is defined in terms of an additional distribution $H_X$ as follows:

$$G_X \sim C_X(H_X)$$
$$H_X = \sum_{X \to Y_1 \dots Y_m \in R_X} \theta_{X \to Y_1 \dots Y_m} \mathrm{TD}_X(G_{Y_1}, \dots, G_{Y_m})$$

That is, the distribution $G_X$ associated with an adapted nonterminal $X \in A$ is a sample from "adapting" (i.e., applying $C_X$ to) its "ordinary" PCFG distribution $H_X$.

Just as with the PCFG, an AG generates the distribution over trees $G_S$, where $S \in N$ is the start symbol. However, while $G_S$ in a PCFG is a fixed distribution (given the rule probabilities $\theta$), in an AG the distribution $G_S$ is itself a random variable (because each $G_X$ for $X \in A$ is random).

Informally, an AG can be understood as caching the trees associated with adapted nonterminals. Generating a tree associated with an adapted nonterminal involves either reusing an already generated tree from the cache, or else generating a "fresh" tree as in a PCFG.

### 3.4 Word segmentation with adaptor grammars

AGs can be used as models of word segmentation, which we briefly review here; see Johnson [12] for more details. The input to the AG consists of a corpus of phoneme strings. For example, the phoneme string corresponding to *Is that the pig?* (with its correct segmentation indicated in blue) is as follows:

$$\text{i } {}_{\vartriangle}\text{z} {}_{\blacktriangle}\text{D} {}_{\vartriangle}\text{\&} {}_{\vartriangle}\text{t} {}_{\blacktriangle}\text{D} {}_{\vartriangle}\text{e} {}_{\blacktriangle}\text{p} {}_{\vartriangle}\text{I} {}_{\vartriangle}\text{g}$$

We can represent any possible segmentation of any possible sentence as a tree generated by the following *unigram* AG.

$$\begin{aligned}
&\text{Sentence} \to \underline{\text{Word}}^+ \\
&\underline{\text{Word}} \to \text{Phoneme}^+ \\
&\text{Phonemes} \to a \mid b \mid \dots
\end{aligned} \tag{4}$$

The trees generated by this adaptor grammar are the same as the trees generated by the CFG rules. (In this and following grammars, the Kleene "+" is expanded into a set of left-recursive rules). For example, the following skeletal parse in which all but the Word nonterminals are suppressed (the others are deterministically inferrable) shows the parse that corresponds to the correct segmentation of the string above.

(Word i z) (Word D & t) (Word D e) (Word p I g)

Because the Word nonterminal in the AG is *adapted* (indicated here by underlining) the adaptor grammar learns the probability of the entire Word subtrees (e.g., the probability that *pIg* is a Word); see [12] for further details. This AG implements the unigram segmentation model of Goldwater et al [9], and as explained in section 2, it has the same tendancy to undersegment as the original unigram model.

The *collocation* AG (5) produces a more accurate segmentation because it models (and therefore "explain away") some of the inter-word dependencies.

$$\begin{aligned}
&\text{Sentence} \to \underline{\text{Colloc}}^+ \\
&\underline{\text{Colloc}} \to \underline{\text{Word}}^+ \\
&\underline{\text{Word}} \to \text{Phoneme}^+ \\
&\text{Phonemes} \to a \mid b \mid \dots
\end{aligned} \tag{5}$$

The collocation AG is a hierarchical process, where the base distribution for the Colloc (collocation) nonterminal adaptor is generated from the Word distribution. The collocation AG generates a sentence as a sequence of Colloc (collocation) nonterminals, each of which is a sequence of Word nonterminals. It generates skeletal parses such as the following:

(Colloc (Word i z)) (Colloc (Word D & t)) (Colloc (Word D e) (Word p I g))

In this parse, *iz* and *D&t* are analysed as both Words and Collocations, while *De pIg* is analysed as a Collocation consisting of two Words. Given training corpora like the ones we use here, the collocations this AG finds are often noun phrases.

### 3.5 Adaptor grammars for joint segmentation and word-object acquisition

This section explains how to combine the word-object reference PCFGs presented in section 3.2 with the word segmentation AGs presented in section 3.4. Combining the word-object reference PCFGs (2) or (3) with the unigram AG (4) is relatively straight-forward; all we need to do is replace the last rule $B_o \rightarrow w$ in these grammars with $\underline{B_o} \rightarrow \text{Phoneme}^+$, i.e., the $B_o$ nonterminals expand to an arbitray sequence of phonemes, and the $\overline{B_o}$ nonterminals are adapted, so these subtrees are cached and reused as appropriate. For example, the *unigram-reference* AG is as follows:

$$
\begin{array}{ll}
\text{S} \rightarrow A_o & o \in O' \\
A_o \rightarrow c & c \in C, o \in c \\
A_o \rightarrow A_o\,B_o & o \in O' \\
A_o \rightarrow A_o\,B_\emptyset & o \in O' \\
\underline{B_o} \rightarrow \text{Phoneme}^+ & o \in O'
\end{array}
$$

The unigram-reference AG specifies essentially the same model as the one investigated in Jones et al [8], and the results below are consistent with those that Jones et al report. This grammar generates a skeletal parses such as the following:

$$(\text{B}_\emptyset \text{ i z}) (\text{B}_\emptyset \text{ D \& t}) (\text{B}_\emptyset \text{ D e}) (\text{B}_{\text{PIG}} \text{ p I g})$$

The *unigram-reference1* AG is similiar to the unigram-reference AG, except that it stipulates that at most one word per sentence is associated with a (non-null) object.

It is also possible to combine the word-object reference PCFGs with the collocation AG. The resulting AGs are straight-forward but more complex, so they are not shown here. The *collocation-reference* AG is a combination of the collocation AG for word segmentation and the reference PCFG for modelling the word-object relationship. It permits an arbitrary number of words in a sentence to be referential.

Interestingly, there are two different reasonable ways of combining the collocation AG with the reference1 PCFG. The *collocation-reference1* AG requires that at most one word in a sentence is referential, just like the reference1 PCFG (3).

The *collocation-referenceC1* AG is similiar to the collocation-reference1 AG, except that it requires that at most one word *in a collocation* is referential. This means that the collocation-referenceC1 AG permits multiple referential words in a sentence (but they must all refer to the same object). This AG is linguistically plausible because a collocation often consists of a content word, which may be referential, surrounded by function words, which are generally not referential.

## 4 Experimental results

We used the same training corpus as Jones et al [8], which was based on the corpus collected by Fernald et al [19] annotated with the objects in the non-linguistic context by Frank et al [7]. In these experiments we used the publically-available AG inference software described in [15]. Rather than specifying the concentration parameters of each Pitman-Yor Processes (PYPs) associated with the adapted nonterminals, that software permits us to place priors on them and sample them. Here we placed a uniform prior on all PYP $a$ parameters and a sparse $\text{Gamma}(100, 0.01)$ prior on the PYP $b$ parameters.

For each grammar we ran 8 MCMC chains for 5,000 iterations each over the corpus, and collected the sample parses from every 10th iteration from the last 2,500 iterations generated by each run. For each sentence in each sample we extracted the word segmentation and the word-object relationships the parse implies, so we obtained 2,000 sample analyses for each sentence in the corpus. We computed the modal (i.e., most frequent) analysis of each sentence, and this is what we scored below [15].

Perhaps the most basic question is: *does non-linguistic context help word segmentation?* We measure accuracy here by token f-score [9]. Jones et al [8] investigated this question by comparing analyses from what we are calling the unigram and unigram-reference models, and failed to find any overall effect of the non-linguistic context (although they did show that it improves the segmentation accuracy of referential words). However, as the following table shows, we do see a marked

improvement in word segmentation f-score when we combine non-linguistic context with the more accurate collocation models.

| Model | word segmentation f-score |
|---|---|
| unigram | 0.533 |
| unigram-reference | 0.537 |
| unigram-reference1 | 0.547 |
| collocation | 0.695 |
| collocation-reference | 0.726 |
| collocation-reference1 | 0.719 |
| collocation-referenceC1 | **0.750** |

We can also ask the converse question: *does better word segmentation improve sentence referent identification?* Here we measure how well the models identify which object, if any, this sentence refers to, and does not directly evaluate word segmentation accuracy. The baseline model here assigns each sentence the "null" $\emptyset$ object, achieving an accuracy of 0.709. As the table below shows, only the collocation-referenceC1 AG with its more complex constraints on the word-object relationship clearly surpasses this baseline. We can also measure the f-score with which the models identify non-$\emptyset$ sentence referents; now the trivial baseline model achieves 0 f-score.

| Model | sentence referent accuracy | sentence referent f-score |
|---|---|---|
| unigram | 0.709 | 0 |
| unigram-reference | 0.702 | 0.355 |
| unigram-reference1 | 0.503 | 0.495 |
| collocation | 0.709 | 0 |
| collocation-reference | 0.728 | 0.280 |
| collocation-reference1 | 0.440 | 0.493 |
| collocation-referenceC1 | **0.839** | **0.747** |

We see a marked improvement in sentence referent accuracy and sentence referent f-score with the collocation-referenceC1 AG.

Finally, we can ask: *how well do the models identify the head nouns of referring noun phrases*, such as *pIg* in *De pIg*? We measure this by calculating the f-score of (word,object) token pairs identified by the model, where the object is not $\emptyset$. This is a single number that indicates how good the models are at identifying referring words and the words that they refer to.

| Model | topical word f-score |
|---|---|
| unigram | 0 |
| unigram-reference | 0.149 |
| unigram-reference1 | 0.147 |
| colloc | 0 |
| collocation-reference | 0.220 |
| collocation-reference1 | 0.321 |
| collocation-referenceC1 | **0.636** |

Again, we find that the collocation-referenceC1 AG identifies referring words and the objects they refer to more accurately than the other models.

## 5 Conclusion

This paper has used Adaptor Grammars (AGs) to formulate a variety of models that jointly segment utterances into words and identify the objects in the non-linguistic context that some of these words refer to. The AGs differed in the kinds of generalisations they are capable of learning, and in the relationship between word segmentation and word reference that they assume. The most accurate results in word segmentation and in the identification of the word-object relationship were obtained by the collocation-referenceC1 AG that tightly integrates a collocation-based model of word segmentation with constraints that require no more than one referential word per collocation. As argued in the introduction, this is consistent with an "interactive" approach to language learning.

# References

[1] Patricia K. Kuhl. Early language acquisition: Cracking the speech code. *Nature Reviews Neuroscience*, 5:831–843, 2004.

[2] Katharine Graf Estes, Julia L. Evans, Martha W. Alibali, and Jenny R. Saffran. Can infants map meaning to newly segmented words? statistical segmentation and word learning. *Psychological Science*, 18(3):254–260, 2007.

[3] James L. McClelland and David E. Rummelhart. An interactive activation model of context effects in letter perception. *Psychological Review*, 88(5):375–407, 1981.

[4] Jeffrey Elman. Finding structure in time. *Cognitive Science*, 14:197–211, 1990.

[5] M. Brent and T. Cartwright. Distributional regularity and phonotactic constraints are useful for segmentation. *Cognition*, 61:93–125, 1996.

[6] M. Brent. An efficient, probabilistically sound algorithm for segmentation and word discovery. *Machine Learning*, 34:71–105, 1999.

[7] Michael C. Frank, Noah Goodman, and Joshua Tenenbaum. Using speakers' referential intentions to model early cross-situational word learning. *Psychological Science*, 20:579–585, 2009.

[8] Bevan K. Jones, Mark Johnson, and Michael C. Frank. Learning words and their meanings from unsegmented child-directed speech. In *Human Language Technologies: The 2010 Annual Conference of the North American Chapter of the Association for Computational Linguistics*, pages 501–509, Los Angeles, California, June 2010. Association for Computational Linguistics.

[9] Sharon Goldwater, Thomas L. Griffiths, and Mark Johnson. A Bayesian framework for word segmentation: Exploring the effects of context. *Cognition*, 112(1):21 – 54, 2009.

[10] Y. W. Teh, M. Jordan, M. Beal, and D. Blei. Hierarchical Dirichlet processes. *Journal of the American Statistical Association*, 101:1566–1581, 2006.

[11] Mark Johnson, Thomas L. Griffiths, and Sharon Goldwater. Adaptor Grammars: A framework for specifying compositional nonparametric Bayesian models. In B. Schölkopf, J. Platt, and T. Hoffman, editors, *Advances in Neural Information Processing Systems 19*, pages 641–648. MIT Press, Cambridge, MA, 2007.

[12] Mark Johnson. Using adaptor grammars to identifying synergies in the unsupervised acquisition of linguistic structure. In *Proceedings of the 46th Annual Meeting of the Association of Computational Linguistics*, Columbus, Ohio, 2008. Association for Computational Linguistics.

[13] David M. Blei, Andrew Y. Ng, and Michael I. Jordan. Latent Dirichlet allocation. *Journal of Machine Learning Research*, 3:993–1022, 2003.

[14] Mark Johnson. PCFGs, topic models, adaptor grammars and learning topical collocations and the structure of proper names. In *Proceedings of the 48th Annual Meeting of the Association for Computational Linguistics*, pages 1148–1157, Uppsala, Sweden, July 2010. Association for Computational Linguistics.

[15] Mark Johnson and Sharon Goldwater. Improving nonparameteric Bayesian inference: experiments on unsupervised word segmentation with adaptor grammars. In *Proceedings of Human Language Technologies: The 2009 Annual Conference of the North American Chapter of the Association for Computational Linguistics*, pages 317–325, Boulder, Colorado, June 2009. Association for Computational Linguistics.

[16] Shay B. Cohen, David M. Blei, and Noah A. Smith. Variational inference for adaptor grammars. In *Human Language Technologies: The 2010 Annual Conference of the North American Chapter of the Association for Computational Linguistics*, pages 564–572, Los Angeles, California, June 2010. Association for Computational Linguistics.

[17] Kenichi Kurihara and Taisuke Sato. Variational Bayesian grammar induction for natural language. In *8th International Colloquium on Grammatical Inference*, 2006.

[18] Mark Johnson, Thomas Griffiths, and Sharon Goldwater. Bayesian inference for PCFGs via Markov chain Monte Carlo. In *Human Language Technologies 2007: The Conference of the North American Chapter of the Association for Computational Linguistics; Proceedings of the Main Conference*, pages 139–146, Rochester, New York, April 2007. Association for Computational Linguistics.

[19] Anne Fernald and Hiromi Morikawa. Common themes and cultural variations in Japanese and American mothers' speech to infants. *Child Development*, 64(3):637–656, 1993.

